# Optimal information decoding from neuronal populations with specific stimulus selectivity

**Marcelo A. Montemurro**
The University of Manchester
Faculty of Life Sciences
Moffat Building
PO Box 88, Manchester M60 1QD, UK
m.montemurro@manchester.ac.uk

**Stefano Panzeri** *
The University of Manchester
Faculty of Life Sciences
Moffat Building
PO Box 88, Manchester M60 1QD, UK
s.panzeri@manchester.ac.uk

## Abstract

A typical neuron in visual cortex receives most inputs from other cortical neurons with a roughly similar stimulus preference. Does this arrangement of inputs allow efficient readout of sensory information by the target cortical neuron? We address this issue by using simple modelling of neuronal population activity and information theoretic tools. We find that efficient synaptic information transmission requires that the tuning curve of the afferent neurons is approximately as wide as the spread of stimulus preferences of the afferent neurons reaching the target neuron. By meta analysis of neurophysiological data we found that this is the case for cortico-cortical inputs to neurons in visual cortex. We suggest that the organization of V1 cortico-cortical synaptic inputs allows optimal information transmission.

## 1 Introduction

A typical neuron in visual cortex receives most of its inputs from other visual cortical neurons. The majority of cortico-cortical inputs arise from afferent cortical neurons with a preference to stimuli which is similar to that of the target neuron [1, 2, 3]. For example, orientation selective neurons in superficial layers in ferret visual cortex receive more than 50% of their cortico-cortical excitatory inputs from neurons with orientation preference which is less than $30^o$ apart. However, this input structure is rather broad in terms of stimulus-specificity: cortico-cortical connections between neurons tuned to dissimilar stimulus orientation also exist [4]. The structure and spread of the stimulus specificity of cortico-cortical connections has received a lot of attention because of its importance for understanding the mechanisms of generation of orientation tuning (see [4] for a review). However, little is still known on whether this structure of inputs allows efficient transmission of sensory information across cortico-cortical synapses.

It is likely that efficiency of information transmission across cortico-cortical synapses also depends on the width of tuning curves of the afferent cortical neurons to stimuli. In fact, theoretical work on population coding has shown that the width of the tuning curves has

---

an important influence on the quality and the nature of the information encoding in cortical populations [5, 6, 7, 8]. Another factor that may influence the efficiency of cortico-cortical synaptic information transmission is the biophysical capability of the target neuron. To conserve all information during synaptic transmission, the target neuron must conserve the 'label' of the spikes arriving from multiple input neurons at different places on its dendritic tree [9]. Because of biophysical limitations, a target neuron that e.g. can only sum inputs at the soma may lose a large part of the information present in the afferent activity. The optimal arrangement of cortico-cortical synapses may also depend on the capability of postsynaptic neurons in processing separately spikes from different neurons.

In this paper, we address the problem of whether cortico-cortical synaptic systems encode information efficiently. We introduce a simple model of neuronal information processing that takes into account both the selective distribution of stimulus preferences typical of cortico-cortical connections and the potential biophysical limitations of cortical neurons. We use this model and information theoretic tools to investigate whether there is an optimal trade-off between the spread of distribution of stimulus preference across the afferent neurons and the tuning width of the afferent neurons itself. We find that efficient synaptic information transmission requires that the tuning curve of the afferent neurons is approximately as wide as the spread of stimulus preferences of the afferent fibres reaching the target neuron. By reviewing anatomical and physiological data, we argue that this optimal trade-off is approximately reached in visual cortex. These results suggest that neurons in visual cortex are wired to decode optimally information from a stimulus-specific distribution of synaptic inputs.

## 2 Model of the activity of the afferent neuronal population

We consider a simple model for the activity of the afferent neuronal population based on the known tuning properties and spatial and synaptic organisation of sensory areas.

### 2.1 Stimulus tuning of individual afferent neurons

We assume that the the population is made of a large number $N$ of neurons (for a real cortical neuron, the number $N$ of afferents is in the order of few thousands [10]). The response of each neuron $r_k (k = 1, \cdots, N)$ is quantified as the number of spikes fired in a salient post-stimulus time window of a length $\tau$. Thus, the overall neuronal population response is represented as a spike count vector $\mathbf{r} = (r_1, \cdots, r_N)$.

We assume that the neurons are tuned to a small number $D$ of relevant stimulus parameters [11, 12], such as *e.g.* orientation, speed or direction of motion of a visual object. The stimulus variable will thus be described as a vector $\mathbf{s} = (s_1, \ldots, s_D)$ of dimension $D$. The number of stimulus features that are encoded by the neuron will be left as a free parameter to be varied within the range 1-5 for two reasons. First, although there is evidence that the number of stimulus features encoded by a single neuron is limited [11, 12], more research is still needed to determine exactly how many stimulus parameters are encoded in different areas. Second, a previous related study [8] has shown that, when considering large neuronal populations with a uniform distribution of stimulus preferences (such as an hypercolumn in V1 containing all stimulus orientations) the tuning width of individual neurons which is optimal for population coding depends crucially on the number of stimulus features being encoded. Thus, it is interesting to investigate how the optimal arrangement of cortico-cortical synaptic systems depends on the number of stimulus features being encoded.

The neuronal tuning function of the $k - th$ neuron ($k = 1, \cdots, N$), which quantifies the mean spike count of the $k - th$ neuron to the presented stimulus, is modelled as a Gaussian distribution, characterised by the following parameters: preferred stimulus $\mathbf{s}^{(k)}$, tuning

width $\sigma_f$, and response modulation $m$:

$$f^{(k)}(\mathbf{s}) = m e^{-\frac{(\mathbf{s}-\mathbf{s}^{(k)})^2}{2\sigma_f{}^2}} \qquad (1)$$

The Gaussian tuning curve is a good description of the tuning properties of e.g. V1 or MT neurons to variables such as stimulus orientation motion direction [13, 14, 15], and is hence widely used in models of sensory coding [16, 17]. Large values of $\sigma_f$ indicate coarse coding, whereas small values of $\sigma_f$ indicate sharp tuning.

Spike count responses of each neuron on each trial are assumed to follow a Poisson distribution whose mean is given by the above neuronal tuning function (Eq. 1). The Poisson model is widely used because it is the simplest model of neuronal firing that captures the salient property of neuronal firing that the variance of spike counts is proportional to its mean. The Poisson model neglects all correlations between spikes. This assumption is certainly a simplification but it is sufficient to account for the majority of the information transmitted by real cortical neurons [18, 19, 20] and, as we shall see later, it is mathematically convenient because it makes our model tractable.

### 2.2  Distribution of stimulus preferences among the afferent population

Neurons in sensory cortex receive a large number of inputs from other neurons with a variety of stimulus preferences. However, the majority of their inputs come from neurons with roughly similar stimulus preference [1, 2, 3]. To characterise correctly this type of spread of stimulus preference among the afferent population, we assume (unlike in previous studies [8]), that the probability distribution of the preferred stimulus among afferent neurons follows a Gaussian distribution:

$$P(\hat{\mathbf{s}}) = \frac{1}{(2\pi)^{D/2}\sigma_p^D} e^{-\frac{(\hat{\mathbf{s}}-\hat{\mathbf{s}}_0)^2}{2\sigma_p^2}} \qquad (2)$$

In Eq. (2) the parameter $\hat{\mathbf{s}}_0$ represents the the center of the distribution, thus being the most represented preferred stimulus in the population. (we set, without loss of generality, $\hat{\mathbf{s}}_0 = 0$.) The parameter $\sigma_p$ controls the spread of stimulus preferences of the afferent neuronal population: a small value of $\sigma_p$ indicates that a large fraction of the population have similar stimulus preferences, and a large value of $\sigma_p$ indicates that all stimuli are represented similarly. A Gaussian distribution of stimulus preferences of the afferent population fits well empirical data on distribution of preferred orientations of synaptic inputs of neurons in both deep and superficial layers of ferret primary visual cortex [3].

## 3  Width of tuning and spread of stimulus preferences in visual cortex

To estimate the width of tuning $\sigma_f$ and the spread of stimulus preferences $\sigma_p$ of cortico-cortical afferent populations in visual cortex, we reviewed critically published anatomical and physiological data. We concentrated on excitatory synaptic inputs, which form the majority of inputs to a cortical pyramidal neuron [10]. We computed $\sigma_p$ by fitting (by a least square method) the published histograms of synaptic connections as function of stimulus preference of the input neuron to Gaussian distributions. Similarly, we determined $\sigma_f$ by fitting spike count histograms to Gaussian tuning curves.

When considering a target neuron in ferret primary visual cortex and using orientation as the stimulus parameters, the spread of stimulus preferences $\sigma_p$ of its inputs is $\approx 20^o$ for layer 5/6 neurons [3], and $16^o$ [3] to $23^o$ [21] for layer 2/3 neurons. The orientation tuning width $\sigma_f$ of the cortical inputs to the V1 target neuron is that of other V1 neurons that project to it. This $\sigma_f$ is $17^o$ for Layer 4 neurons [22], and it is similar for neurons in deep and superficial layers [3]. When considering a target neuron in Layer 4 of cat visual cortex

and orientation tuning, the spread of stimulus preference $\sigma_p$ is $20^o$ [2] and $\sigma_f$ is $\approx 17^o$. When considering a target neuron in ferret visual cortex and motion direction tuning, the spread of tuning of its inputs $\sigma_p$ is $\approx 30^o$ [1]. Motion direction tuning widths of macaque neurons is $\approx 28^o$, and this width is similar across species (see [13]).

The most notable finding of our meta-analysis of published data is that $\sigma_p$ and $\sigma_f$ appear to be approximately of the same size and their ratio $\sigma_f/\sigma_p$ is distributed around 1, in the range 0.7 to 1.1 for the above data. We will use our model to understand whether this range of $\sigma_f/\sigma_p$ corresponds to an optimal way to transmit information across a synaptic system.

## 4 Information theoretic quantification of population decoding

To characterise how a target neuronal system can decode the information about sensory stimuli contained in the activity of its afferent neuronal population, we use mutual information [23]. The mutual information between a set of stimuli and the neuronal responses quantifies how well any decoder can discriminate among stimuli by observing the neuronal responses. This measure has the advantage of being independent of the decoding mechanism used, and thus puts precise constraints on the information that can be decoded by any biological system operating on the afferent activity.

Previous studies on the information content of an afferent neuronal population [7, 8] have assumed that the target neuronal decoding system can extract all the information during synaptic transmission. To do so, the target neuron must conserve the "label" of the spikes arriving from multiple neurons at different sites on its dendritic tree [9]. Given the potential biophysical difficulty in processing each spike separately, a simple alternative to spike labelling has been proposed, - spike pooling [10, 24]. In this scheme, the target neuron simply sums up the afferent activity. To characterize how the decoding of afferent information would work in both cases, we compute both the information that can be decoded by either a system that processes separately spikes from different neurons (the "labeled-line information") and the information available to a decoder that sums all incoming spikes (the "pooled information") [9, 24]. In the next two subsections we define these quantities and we explain how we compute it in our model.

### 4.1 The information available to the the labeled-line decoder

The mutual information between the set of the stimuli and the labeled-line neuronal population activity is defined as follows [9, 24]:

$$I^{LL}(\mathbf{S}, \mathbf{R}) = \int d\mathbf{s} P(\mathbf{s}) \sum_{\mathbf{r}} P(\mathbf{r}|s) \log \frac{P(\mathbf{r}|\mathbf{s})}{P(\mathbf{r})} \qquad (3)$$

where $P(\mathbf{s})$ is the probability of stimulus occurrence (here taken for simplicity as a uniform distribution over the hypersphere of $D$ dimensions and 'radius' $s_\rho$). $P(\mathbf{r}|\mathbf{s})$ is the probability of observing a neuronal population response $\mathbf{r}$ conditional to the occurrence of stimulus $\mathbf{s}$, and $P(\mathbf{r}) = \int d\mathbf{s} P(\mathbf{s}) P(\mathbf{r}|\mathbf{s})$. Since the response vector $\mathbf{r}$ keeps separate the spike counts of each neuron, the amount of information in Eq. (3) is only accessible to a decoder than can keep the label of which neuron fired which spike [9, 24]. The probability $P(\mathbf{r}|\mathbf{s})$ is computed according to the Poisson distribution, which is entirely determined by the knowledge of the tuning curves [5]. The labeled-line mutual information is difficult to compute for large populations, because it requires the knowledge of the probability of the large-dimensional response vector $\mathbf{r}$. However, since in our model we assume that we have a very large number of independent neurons in the population and that the total activity of the system is of the order of its size, then we can use the following simpler (but still exact)

expression[16, 25]:

$$I^{LL}(\mathbf{S}, \mathbf{R}) = H(\mathbf{S}) - \frac{D}{2} \ln (2\pi e) + \frac{1}{2} \int d\mathbf{s}\, P(\mathbf{s}) \ln (|\boldsymbol{\mathcal{J}}(\mathbf{s})|) \tag{4}$$

where $H(\mathbf{S})$ is the entropy of the prior stimulus presentation distribution $P(\mathbf{S})$, $\boldsymbol{\mathcal{J}}(\mathbf{s})$ is the Fisher information matrix and $|\dots|$ stands for the determinant. The Fisher information matrix is a $D \times D$ matrix who's elements $i, j$ are defined as follows:

$$\mathcal{J}_{i,j}(\mathbf{s}) = -\sum_{\mathbf{r}} P(\mathbf{r}|\mathbf{s}) \left( \frac{\partial^2}{\partial s_i\, s_j} \log P(\mathbf{r}|\mathbf{s}) \right), \tag{5}$$

Fisher information is a useful measure of the accuracy with which a particular stimulus can be reconstructed from a single trial observation of neuronal population activity. However, in this paper it is used only as a step to obtain a computationally tractable expression for the labeled-line mutual information. The Fisher information matrix can be computed by taking into account that for a population of Poisson neurons is just the sum of the Fisher information for individual neurons, and the latter has a simple expression in terms of tuning curves [16]. Since the neuronal population size $N$ is is large, the sum over Fisher information of individual neurons can be replaced by an integral over the stimulus preferences of the neurons in the population, weighted by their probability density $P(\hat{s})$. After performing the integral over the distribution of preferred stimuli, we arrived at the following result for the elements of the Fisher information matrix:

$$\mathcal{J}_{i,j}(\mathbf{s}) = \frac{N\tau m}{\sigma_p^2} \frac{\sigma^{D-2}}{(1+\sigma^2)^{\frac{D}{2}+2}} \left(\delta_{i,j} + \sigma^2 \left(\delta_{i,j} + \xi_i \xi_j\right)\right) e^{-\frac{\xi^2}{2(1+\sigma^2)}} \tag{6}$$

where we have introduced the following short-hand notation $\sigma_f/\sigma_p \rightarrow \sigma$ and $\mathbf{s}/\sigma_p \rightarrow \boldsymbol{\xi}$; $\delta_{i,j}$ stands for the Kroneker Delta. From Eq. (6) it is possible to compute explicitly the determinant $|\boldsymbol{\mathcal{J}}(\mathbf{s})|$, which has the following form:

$$|\boldsymbol{\mathcal{J}}(\mathbf{s})| = \prod_{i=1}^{D} \lambda_i = \alpha(\xi)^D (1+\sigma^2)^{D-1} \left(1 + \sigma^2 (1+\xi^2)\right) \tag{7}$$

where $\alpha(\xi)$ is given by:

$$\alpha(\xi) = \frac{N\tau m}{\sigma_p^2} \frac{\sigma^{D-2}}{(1+\sigma^2)^{\frac{D}{2}+1}} e^{-\frac{\xi^2}{2(1+\sigma^2)}} \tag{8}$$

Inserting Eq. (7) into Eq. (4), one obtains a tractable but still exact expression for the mutual information , which has the advantage over Eq. (3) of requiring only an integral over a $D$-dimensional stimulus rather than a sum over an infinite population.

We have studied numerically the dependence of the labeled-line information on the parameters $\sigma_f$ and $\sigma_p$ as a function of the number of encoded stimulus features $D$ [1]. We investigated this by fixing $\sigma_p$ and then varying the ration $\sigma_f/\sigma_p$ over a wide range. Results (obtained for $\sigma_p = 1$ but representative of a wide $\sigma_f$ range) are reported in Fig. 1. We found that, unlike the case of a uniform distribution of stimulus preferences [8], there is a finite value of the width of tuning $\sigma_f$ that maximizes the information for all $D \geq 2$. Interestingly, for $D \geq 2$ the range $0.7 \leq \sigma_f/\sigma_p \leq 1.1$ found in visual cortex either contains the maximum or corresponds to near optimal values of information transmission. For $D = 1$, information is maximal for very narrow tuning curves. However, also in this case the information values are still efficient in the cortical range $\sigma_f/\sigma_p \approx 1$, in that the tail of the $D = 1$ information curve is avoided in that region. Thus, the range of values of $\sigma_f$ and $\sigma_p$ found in visual cortex allows efficient synaptic information transmission over a wide range of number of stimulus features encoded by the neuron.

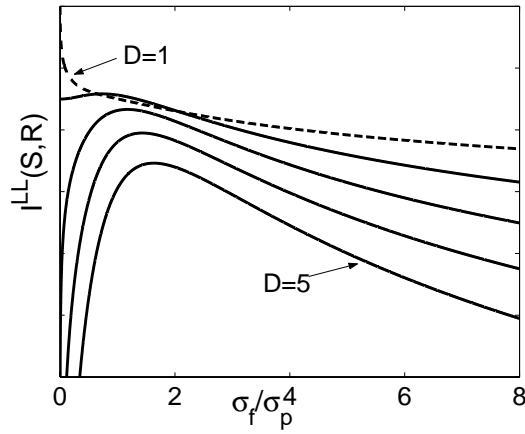

Figure 1: Mutual labeled-line information as a function of the ratio of tuning curve width and stimulus preference spread $\sigma_f/\sigma_p$. The curves for each stimulus dimensionality $D$ were shifted by a constant factor to separate them for visual inspection (lower curves correspond to higher values of $D$). The y-axis is thus in arbitrary units. The position of the maximal information for each stimulus dimension falls either inside the range of values of $\sigma_f/\sigma_p$ found in visual cortex, or very close to it (see text) . Parameters are as follows: $\mathbf{s}_\rho = 2$, $r_{max} = 50$Hz, $\tau = 10$ms.

## 4.2 The information available to the the pooling decoder

We now consider the case in which the target neuron cannot process separately spikes from different neurons (for example, a neuron that just sums up post-synaptic potentials of approximately equal weight at the soma). In this case the label of the neuron that fired each spike is lost by the target neuron, and it can only operate on the pooled neuronal signal, in which the identity of each spike is lost. Pooling mechanisms have been proposed as simple information processing strategies for the nervous system. We now study how pooling changes the requirements for efficient decoding by the target neuron.

Mathematically speaking, pooling maps the vector $\mathbf{r}$ onto a scalar $\rho$ equal to the sum of the individual activities: $\rho = \sum r_k$. Thus, the mutual information that can be extracted by any decoder that only pools it inputs is given by the following expression:

$$I^P(\mathbf{S}, \mathbf{R}) = \int d\mathbf{s} P(\mathbf{s}) \sum_\rho P(\rho|s) \log \frac{P(\rho|\mathbf{s})}{P(\rho)} \tag{9}$$

where $P(\rho|\mathbf{s})$ and $P(\rho)$ are the the stimulus-conditional and stimulus-unconditional probability of observing a pooled population response $\rho$ on a single trial. The probability $P(\rho|\mathbf{s})$ can be computed by noting that a sum of Poisson-distributed responses is still a Poisson-distributed response whose tuning curve to stimuli is just the sum of the individual tuning curves. The pooled mutual information is thus a function of a single Poisson-distributed response variables and can be computed easily also for large populations.

The dependence of the pooled information on the parameters $\sigma_f$ and $\sigma_p$ as a function of the number of encoded stimulus features $D$ is reported in Fig. 2. There is one important difference with respect to the labeled-line information transmission case. The difference is that, for the pooled information, there is a finite optimal value for information transmission also when the neurons are tuned to one-dimensional stimulus feature. For all cases of stimulus dimensionality considered, the efficient information transmission though the pooled

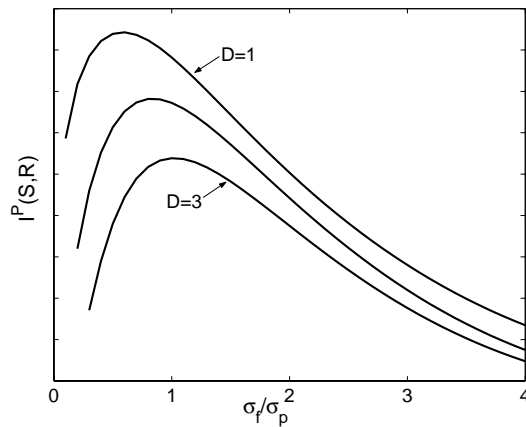

Figure 2: Pooled mutual information as a function of the ratio of tuning curve width and stimulus preference spread $\sigma_f/\sigma_p$. The maxima are inside the range of experimental values of $\sigma_f/\sigma_p$ found in the visual cortex, or very close to it (see text). As for Fig. 1, the curves for each stimulus dimensionality $D$ were shifted by a constant factor to separate them for visual inspection (lower curves correspond to higher values of $D$). The y-axis is thus in arbitrary units. Parameters are as follows: $\mathbf{s}_\rho = 2$, $r_{max} = 50$ Hz,$\tau = 10$ms.

neuronal decoder can still be reached in the visual cortical range $0.7 \leq \sigma_f\ \sigma_p \leq 1.1$. This finding shows that the range of values of $\sigma_f$ and $\sigma_p$ found in visual cortex allows efficient synaptic information transmission even if the target neuron does not rely on complex dendritic processing to conserve the label of the neuron that fired the spike.

## 5  Conclusions

The stimulus specificity of cortico-cortical connections is important for understanding the mechanisms of generation of orientation tuning (see [4]) for a review). Here, we have shown that the stimulus-specific structure of cortico-cortical connections may have also implications for understanding cortico-cortical information transmission. Our results suggest that, whatever the exact role of cortico-cortical synapses in generating orientation tuning, their wiring allows efficient transmission of sensory information.

### Acknowledgments

We thanks A. Nevado and R. Petersen for many interesting discussions. Research supported by ICTP, HFSP, Royal Society and Wellcome Trust 066372/Z/01/Z.

## Footnotes

[1]We found (data not shown) that other parameters such as $m$ and $\tau$, had a weak or null effect on the optimal configuration; see [17] for a $D = 1$ example in a different context.

## References

[1] B. Roerig and J. P. Y. Kao. Organization of intracortical circuits in relation to direction preference maps in ferret visual cortex. *J. Neurosci.*, 19:RC44(105), 1999.

[2] T. Yousef, T. Bonhoeffer, D-S. Kim, U. T. Eysel, É. Tóth, and Z. F. Kisvárday. Orientation topography of layer 4 lateral networks revealed by optical imaging in cat visual cortex (area 18). *European J. Neurosci.*, 11:4291–4308, 1999.

[3] B. Roerig and B. Chen. Relations of local inhibitory and excitatory circuits to orientation preference maps in ferret visual cortex. *Cerebral Cortex*, 12:187–198, 2002.

[4] K. A. C. Martin. Microcircuits in visual cortex. *Current Opinion in Neurobiology*, 12:418–425, 2002.

[5] P. Dayan and L. F. Abbott. *Theoretical Neuroscience*. MIT Press, 2001.

[6] D. C. Fitzpatrick, R. Batra, T. R. Stanford, and S. Kuwada. A neuronal population code for sound localization. *Nature*, 388:871–874, 1997.

[7] A. Pouget, S. Deneve, J-C. Ducom, and P.E. Latham. Narrow versus wide tuning curves: what's best for a population code? *Neural Computation*, 11:85–90, 1999.

[8] K. Zhang and T.J. Sejnowski. Neuronal tuning: to sharpen or to broaden? *Neural Computation*, 11:75–84, 1999.

[9] D. S. Reich, F. Mechler, and J. D. Victor. Independent and redundant information in nearby cortical neurons. *Science*, 294:2566–2568, 2001.

[10] M. N. Shadlen and W. T. Newsome. The variable discharge of cortical neurons: implications for connectivity, computation and coding. *J. Neurosci.*, 18(10):3870–3896, 1998.

[11] N. Brenner, W. Bialek, and R. de Ruyter van Steveninck. Adaptive rescaling maximizes information transmission. *Neuron*, 26:695–702, 2000.

[12] J. Touryan, B. Lau, and Y. Dan. Isolation of relevant visual features from random stimuli for cortical complex cells. *J. Neurosci*, 22:10811–10818, 2002.

[13] T. D. Albright. Direction and orientation selectivity of neurons in visual area MT of the macaque. *J. Neurophysiol.*, 52:1106–1130, 1984.

[14] K.H. Britten, M. N. Shadlen, W. T. Newsome, and J. A. Movshon. The analysis of visual-motion - a comparison of neuronal and psychophysical performance. *J. Neurosci.*, 12:4745–4765, 1992.

[15] K Kang, RM Shapley, and H Sompolinsky. Information tuning of population of neurons in primary visual cortex. *J. Neurosci.*, 24:3726–3735, 2004.

[16] N. Brunel and J. P. Nadal. Mutual information, fisher information and population coding. *Neural Computation*, 10:1731–1757, 1998.

[17] A. Nevado, M.P. Young, and S. Panzeri. Functional imaging and neural information coding. *Neuroimage*, 21:1095–1095, 2004.

[18] S. Nirenberg, S. M. Carcieri, A.L. Jacobs, and P. E. Latham. Retinal ganglion cells act largely as independent encoders. *Nature*, 411:698–701, 2001.

[19] R. S. Petersen, S. Panzeri, and M.E. Diamond. Population coding of stimulus location in rat somatosensory cortex. *Neuron*, 32:503–514, 2001.

[20] M. W. Oram, N.G. Hatsopoulos, B.J. Richmond, and J.P. Donoghue. Excess synchrony in motor cortical neurons provides redundant direction information with that from coarse temporal measures. *J. Neurophysiol.*, 86:1700–1716, 2001.

[21] M. B. Dalva, M. Weliky, and L. Katz. Relations between local synaptic connections and orientation domains in primary visual cortex. *Neuron*, 19:871–880, 1997.

[22] W. M. Usrey, M. P. Sceniak, and B. Chapman. Receptive fields and response properties of neurons in layer 4 of ferret visual cortex. *J. Neurophysiol.*, 89:1003–1015, 2003.

[23] T.M. Cover and J.A. Thomas. *Elements of information theory*. John Wiley, 1991.

[24] S. Panzeri, F. Petroni, R.S. Petersen, and M.E. Diamond. Decoding neuronal population activity in rat somatosensory cortex: role of columnar organization. *Cerebral Cortex*, 13:45–52, 2003.

[25] K Kang and H Sompolinsky. Mutual information of population codes and distance measures in probability space. *Phys. Rev. Lett.*, 86:4958–4961, 2001.
